# Memory Capacity of Linear vs. Nonlinear Models of Dendritic Integration

**Panayiota Poirazi***
Biomedical Engineering Department
University of Southern California
Los Angeles, CA 90089
*poirazi@scf.usc.edu*

**Bartlett W. Mel***
Biomedical Engineering Department
University of Southern California
Los Angeles, CA 90089
*mel@lnc.usc.edu*

## Abstract

Previous biophysical modeling work showed that nonlinear interactions among nearby synapses located on active dendritic trees can provide a large boost in the memory capacity of a cell (Mel, 1992a, 1992b). The aim of our present work is to quantify this boost by estimating the capacity of (1) a neuron model with passive dendritic integration where inputs are combined linearly across the entire cell followed by a single global threshold, and (2) an active dendrite model in which a threshold is applied separately to the output of each branch, and the branch subtotals are combined linearly. We focus here on the limiting case of binary-valued synaptic weights, and derive expressions which measure model capacity by estimating the number of distinct input-output functions available to both neuron types. We show that (1) the application of a fixed nonlinearity to each dendritic compartment substantially increases the model's flexibility, (2) for a neuron of realistic size, the capacity of the nonlinear cell can exceed that of the same-sized linear cell by more than an order of magnitude, and (3) the largest capacity boost occurs for cells with a relatively large number of dendritic subunits of relatively small size. We validated the analysis by empirically measuring memory capacity with randomized two-class classification problems, where a stochastic delta rule was used to train both linear and nonlinear models. We found that large capacity boosts predicted for the nonlinear dendritic model were readily achieved in practice.

*http://lnc.usc.edu

# 1  Introduction

Both physiological evidence and connectionist theory support the notion that in the brain, memories are stored in the pattern of learned synaptic weight values. Experiments in a variety of neuronal preparations however, indicate that the efficacy of synaptic transmission can undergo substantial fluctuations up or down, or both, during brief trains of synaptic stimuli. Large fluctuations in synaptic efficacy on short time scales seem inconsistent with the conventional connectionist assumption of stable, high-resolution synaptic weight values. Furthermore, a recent experimental study suggests that excitatory synapses in the hippocampus—a region implicated in certain forms of explicit memory—may exist in only a few long-term stable states, where the continuous grading of synaptic strength seen in standard measures of long-term potentiation (LTP) may exist only in the average over a large population of two-state synapses with randomly staggered thresholds for learning (Petersen, Malenka, Nicoll, & Hopfield, 1998). According to conventional connectionist notions, the possibility that individual synapses hold only one or two bits of long-term state information would seem to have serious implications for the storage capacity of neural tissue. Exploration of this question is one of the main themes of this paper.

In a related vein, we have found in previous biophysical modeling studies that nonlinear interactions between synapses co-activated on the same branch of an active dendritic tree could provide an alternative form of long-term storage capacity. This capacity, which is largely orthogonal to that tied up in conventional synaptic weights, is contained instead in the spatial permutation of synaptic connections onto the dendritic tree—which could in principle be modified in the course of learning or development (Mel, 1992a, 1992b). In a more abstract setting, we recently showed that a large repository of model flexibility lies in the *choice* as to which of a large number of possible interaction terms available in high dimension is actually included in a learning machine's discriminant function, and that the excess capacity contained in this "choice flexibility" can be quantified using straightforward counting arguments (Poirazi & Mel, 1999).

# 2  Two Alternative Models of Dendritic Integration

In this paper, we use a similar function-counting approach to address the more biologically relevant case of a neuron with multiple quasi-independent dendritic compartments (fig. 1). Our primary objective has been to compare the memory capacity of a cell assuming two different modes of dendritic integration. According to the linear model, the neuron's activation level $a_L(\mathbf{x})$ prior to thresholding is given by a weighted sum of of its inputs over the cell as a whole. According to the nonlinear model, the $k$ synaptic inputs to each branch are first combined linearly, a static (e.g. sigmoidal) nonlinearity is applied to each of the $m$ branch subtotals, and the resulting branch outputs are summed to produce the cell's overall activity $a_N(\mathbf{x})$:

$$a_L(\mathbf{x}) = \sum_{i=1}^{m} \left( \sum_{j=1}^{k} x_j^{(i)} \right) \qquad\qquad a_N(\mathbf{x}) = \sum_{i=1}^{m} g\left( \sum_{j=1}^{k} x_j^{(i)} \right) \qquad (1)$$

The expressions for $a_L$ and $a_N$ were written in similar form to emphasize that the models have an identical number of synaptic weights, differing only in the presence or absence of a fixed nonlinear function $g$ applied to the branch subtotals. Though individual synaptic weights in both models are constrained to have a value of 1, any of the $d$ input lines may form multiple connections on the same or different

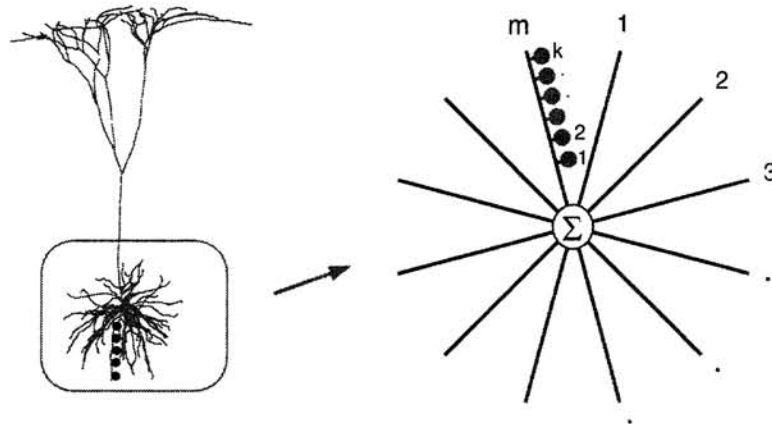

Figure 1: A cell is modeled as a set of $m$ identical branches connected to a soma, where each branch contains $k$ synaptic contacts driven by one of $d$ distinct input lines.

branches as a means of representing graded synaptic strengths. Similarly, an input line which forms no connection has an implicit weight of 0. In light of this restriction to positive (or zero) weight values, both the linear and nonlinear models are split into two opponent channels $a^+$ and $a^-$ dedicated to positive vs. negative coefficients, respectively. This leads to a final output for each model:

$$y_L(\mathbf{x}) = sgn\ [a_L^+(\mathbf{x}) - a_L^-(\mathbf{x})] \qquad y_N(\mathbf{x}) = sgn\ [a_N^+(\mathbf{x}) - a_N^-(\mathbf{x})] \qquad (2)$$

where the *sgn* operator maps the total activation level into a class label of $\{-1, 1\}$.

In the following, we derive expressions for the number of distinct parameter states available to the linear vs. nonlinear models, a measure which we have found to be a reliable predictor of storage capacity under certain restrictions (Poirazi & Mel, 1999). Based on these expressions, we compute the capacity boost provided by the branch nonlinearity as a function of the number of branches $m$, synaptic sites per branch $k$, and input space dimensionality $d$. Finally, we test the predictions of the analytical model by training both linear and nonlinear models on randomized classification problems using a stochastic delta rule, and empirically measure and compare the storage capacities of the two models.

## 3 Results

### 3.1 Counting Parameter States: Linear vs. Nonlinear Model

We derived expressions for $B_L$ and $B_N$, which estimate the total number of parameter bits available to the linear vs. nonlinear models, respectively:

$$B_N = 2\log_2 \left( \dbinom{\dbinom{k+d-1}{k} + m - 1}{m} \right) \qquad B_L = 2\log_2 \left( \dbinom{s+d-1}{s} \right)$$

$$(3)$$

These expressions estimate the number of non-redundant states in each neuron type, i.e., those assignments of input lines to dendritic sites which yield distinct

input-output functions $y_L$ or $y_N$.

These formulae are plotted in figure 2A with $d = 100$, where each curve represents a cell with a fixed number of branches (indicated by $m$). In each case, the capacity increases steadily as the number of synapses per branch, $k$, is increased. The logarithmic growth in the capacity of the linear model (evident in an asymptotic analysis of the expression for $B_L$) is shown at the bottom of the graph (circles), from which it may be seen that the boost in capacity provided by the dendritic branch nonlinearity increases steadily with the number of synaptic sites. For a cell with 100 branches containing 100 synaptic sites each, the capacity boost relative to the linear model exceeds a factor of 20.

Figure 2B shows that for a given total number of synaptic sites, in this case $s = m \cdot k = 10,000$, the capacity of the nonlinear cell is maximized for a specific choice of $m$ and $k$. The peak of each of the three curves (computed for different values of $d$) occurs for a cell containing 1,250 branches with 8 synapses each. However, the capacity is only moderately sensitive to the branch count: the capacity of a cell with 100 branches of 100 synapses each, for example, lies within a factor of two of the optimal configuration. The linear cell capacities can be found at the far right edge of the plot ($m = 10,000$), since a nonlinear model with one synapse per branch has a number of trainable states identical to that of a linear model.

## 3.2   Validating the Analytical Model

To test the predictions of the analytical model, we trained both linear and nonlinear cells on randomized two-class classification problems. Training samples were drawn from a 40-dimensional spherical Gaussian distribution and were randomly assigned positive or negative labels—in some runs, training patterns were evenly divided between positive and negative labels, with similar results. Each of the 40 original input dimensions was recoded using a set of 10 1-dimensional binary, non-overlapping receptive fields with centers spaced along each dimension such that all receptive fields would be activated equally often. This manipulation mapped the original 40-dimensional learning problem into 400 dimensions, thereby increasing the discriminability of the training samples. The relative memory capacity of linear vs. nonlinear cells was then determined empirically by comparing the number of training patterns learnable at a fixed error rate of 2%.

The learning rule used for both cell types was similar to the "clusteron" learning rule described in (Mel, 1992a), and involved two mechanisms known to contribute to neural development: (1) random activity-independent synapse formation, and (2) activity-dependent synapse stabilization. In each iteration, a set of 25 synapses was chosen at random, and the "worst" synapse was identified based on the correlation over the training set of (i) the input's pre-synaptic activity, (ii) the post-synaptic activity (i.e. the local nonlinear branch response for the nonlinear energy model or a constant of 1 for the linear model), and (iii) a global "delta" signal with a value of 0 if the cell responded correctly to the input pattern, or $\pm 1$ if the cell responded incorrectly. The poorest-performing synapse on the branch was then targeted for replacement with a new synapse drawn at random from the $d$ input lines. The probability that the replacement actually occurred was given by a Boltzmann equation based on the difference in the training set error rates before and after the replacement. A "temperature" variable was gradually lowered over the course of the simulation, which was terminated when no further improvement in error rates was seen.

Results of the learning runs are shown in fig. 3 where the analytical capacity (measured in bits) was scaled to the numerical capacity (measured in training patterns

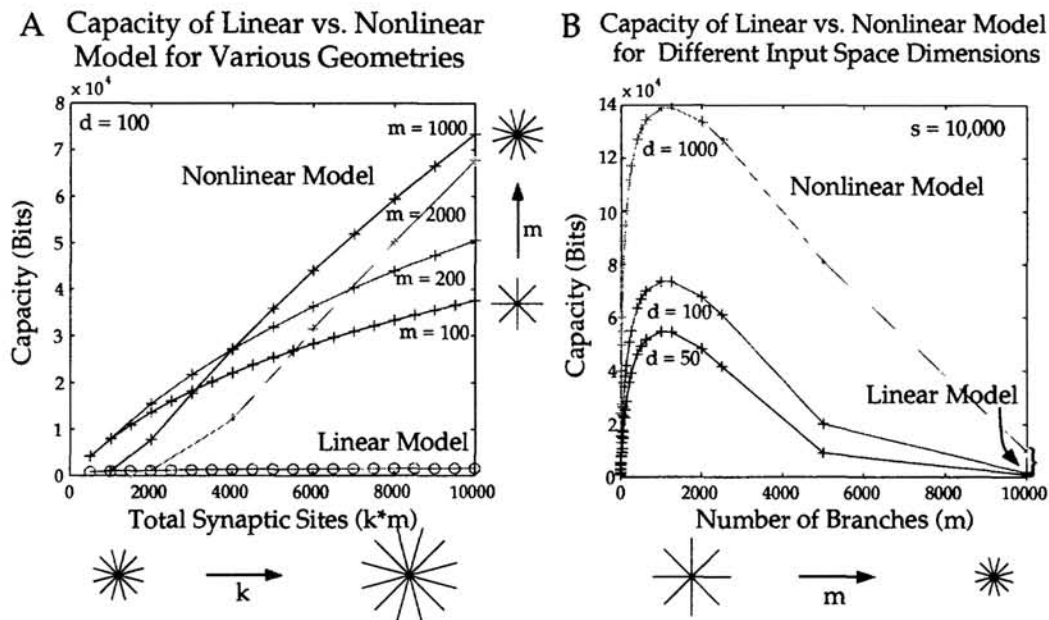

Figure 2: Comparison of linear vs. nonlinear model capacity as a function of branch geometry. A. Capacity in *bits* for linear and several nonlinear cells with different branch counts (for $d = 100$). For each curve indexed by branch count $m$, sites per branch $k$ increases from left to right as indicated iconically beneath the $x$-axis. For all cells, capacity increases with an increasing number of sites, though the capacity of the linear model grows logarithmically, leading to an increasingly large capacity boost for the size-matched nonlinear cells. B. Capacity of a nonlinear model with 10,000 sites for different values of input space dimension $d$. Branch count $m$ grows along the $x$-axis. Cells at right edge of plot contain only one synapse per branch, and thus have a number of modifiable parameters (and hence capacity) equivalent to that of the linear model. All three curves show that there exist an optimal geometry which maximizes the capacity of the nonlinear model (in this case 1,250 branches with 8 synapses each).

learned at 2% error). Two key features of the theoretical curves (dashed lines) are echoed in the empirical performance curves (solid lines), including the much larger storage capacity of the nonlinear cell model, and the specific cell geometry which maximizes the capacity boost.

## 4 Discussion

We found using both analytical and numerical methods that in the limit of low-resolution synaptic weights, application of a fixed output nonlinearity to each compartment of a dendritic tree leads to a significant boost in capacity relative to a cell whose post-synaptic integration is linear. For example, given a cell with 10,000 synaptic contacts originating from 400 distinct input lines, the analysis predicts a 23-fold increase in capacity for the nonlinear cell, while numerical simulations using a stochastic delta rule actually achieve a 15-fold boost.

Given that a linear and a nonlinear model have an identical number of synaptic contacts with uniform synaptic weight values, what accounts for the capacity boost? The principal insight gained in this work is that the attachment of a fixed non-linearity to each branch in a neuron substantially increases its underlying "model

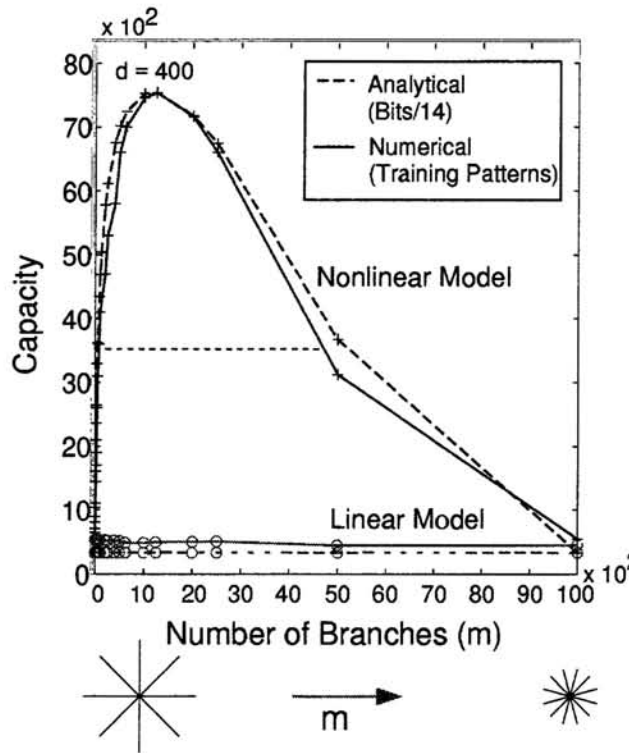

Figure 3: Comparison of capacity boost predicted by analysis vs. that observed empirically when linear and nonlinear models were trained using the same stochastic delta rule. Dashed lines: analytical curves for linear vs. nonlinear model for a cell with 10,000 sites show capacity for varying cell geometries. Solid lines: empirical performance for same two cells at 2% error criterion, using a subunit nonlinearity $g(x) = x^{10}$ (similar results were seen using a sigmoidal nonlinearity, though the parameters of the optimal sigmoid depended on the cell geometry). For both analytical and numerical curves, peak capacity is seen for cell with 1,000 branches (10 synapses per branch). Capacity exceeds that of same-sized linear model by a factor of 15 at the peak, and by more than a factor of 7 for cells ranging from about 3 to 60 synapses per branch (horizontal dotted line).

flexibility", i.e. confers upon the cell a much larger choice of distinct input-output relations from which to select during learning. This may be illustrated as follows. For the linear model, branching structure is irrelevant so that $y_L$ depends only on the *number* of input connections formed from each of the $d$ input lines. All spatial permutations of a set of input connections are thus interchangeable and produce identical cell responses. This massive redundancy confines the capacity of the linear model to grow only logarithmically with an increasing number of synaptic sites (fig. 1A), an unfortunate limitation for a brain in which the formation of large numbers of synaptic contacts between neurons is routine. In contrast, the model with nonlinear subunits contains many fewer redundancies: most spatial permutations of the same set of input connections lead to *non*-identical values of $y_N$, since an input $x$ swapped from branch $b_1$ to branch $b_2$ leads to the elimination of the $k-1$ interaction terms involving $x$ on branch $b_1$ and the creation of $k-1$ new interaction terms on branch $b_2$.

Interestingly, the particular form of the branch nonlinearity has virtually no effect on the capacity of the cell as far as the counting arguments are concerned (though it can have a profound effect on the cell's "representational bias"—see below), since the principal effect of the nonlinearity in our capacity calculations is to break the symmetry among the different branches.

The issue of representational bias is a critical one, however, and must be considered when attempting to predict absolute or relative performance rates for particular classifiers confronted with specific learning problems. Thus, intrinsic differences in the geometry of linear vs. nonlinear discriminant functions mean that the param-

eters available to the two models may be better or worse suited to solve a given learning problem, even if the two models were equated for total parameter flexibility. While such biases are not taken into account in our analysis, they could nonetheless have a substantial effect on measured error rates—and could thus throw a performance advantage to one machine or the other. One danger is that performance differences measured empirically could be misinterpreted as arising from differences in underlying model capacity, when in fact they arise from differential suitability of the two classifiers for the learning problem at hand. To avoid this difficulty, the random classification problems we used to empirically assess memory capacity were chosen to level the playing field for the linear vs. nonlinear cells, since in a previous study we found that the coefficients on linear vs. nonlinear (quadratic) terms were about equally efficient as features for this task. In this way, differences in measured performance on these tasks were primarily attributable to underlying capacity differences, rather than differences in representational bias. This experimental control permitted more meaningful comparisons between our analytical and empirical tests (fig. 3).

The problem of representational bias crops up in a second guise, wherein the analytical expressions for capacity in eq. 1 can significantly overestimate the actual performance of the cell. This occurs when a particular ensemble of learning problems fails to utilize all of the entropy available in the cell's parameter space—for example, by requiring the cell to visit only a small subset of its parameter states relatively often. This invalidates the maximum parameter entropy assumption made in the derivation of eq. 1, so that measured performance will tend to fall below predicted values. The actual performance of either model when confronted with an ensemble of learning problems will thus be determined by (1) the number of trainable parameters available to the neuron (as measured by eq. 1), (2) the suitability of the neuron's parameters for solving the assigned learning problems, and (3) the utilization of parameters, which relates to the entropy in the joint probability of the parameter values averaged over the ensemble of learning problems. In our comparisons here of linear and nonlinear cells, we we have calculated (1), and have attempted to control for (2) and (3).

In conclusion, our results build upon the results of earlier biophysical simulations, and indicate that in the limit of a large number of low-resolution synaptic weights, nonlinear dendritic processing could nonetheless have a major impact on the storage capacity of neural tissue.

# References

Mel, B. W. (1992a). The clusteron: Toward a simple abstraction for a complex neuron. In Moody, J., Hanson, S., & Lippmann, R. (Eds.), *Advances in Neural Information Processing Systems, vol. 4*, pp. 35–42. Morgan Kaufmann, San Mateo, CA.

Mel, B. W. (1992b). NMDA-based pattern discrimination in a modeled cortical neuron. *Neural Comp.*, *4*, 502–516.

Petersen, C. C. H., Malenka, R. C., Nicoll, R. A., & Hopfield, J. J. (1998). All-or-none potentiation and CA3–CA1 synapses. *Proc. Natl. Acad. Sci. USA*, *95*, 4732–4737.

Poirazi, P., & Mel, B. W. (1999). Choice and value flexibility jointly contribute to the capacity of a subsampled quadratic classifier. *Neural Comp., in press.*
